# Object-Based Analog VLSI Vision Circuits

**Christof Koch**
Computation and Neural Systems
California Institute of Technology
Pasadena, CA

**Bimal Mathur, Shih-Chii Liu**
Rockwell International Science Center
Thousand Oaks, CA

**John G. Harris**
MIT Artificial Intelligence Laboratory
Cambridge, MA

**Jin Luo, Massimo Sivilotti**
Tanner Research, Inc.
Pasadena, CA

## Abstract

We describe two successfully working, analog VLSI vision circuits that move beyond pixel-based early vision algorithms. One circuit, implementing the *dynamic wires* model, provides for dedicated lines of communication among groups of pixels that share a common property. The chip uses the dynamic wires model to compute the arclength of visual contours. Another circuit labels all points inside a given contour with one voltage and all other with another voltage. Its behavior is very robust, since small breaks in contours are automatically sealed, providing for *Figure-Ground* segregation in a noisy environment. Both chips are implemented using networks of resistors and switches and represent a step towards object level processing since a single voltage value encodes the property of an ensemble of pixels.

## 1 CONTOUR-LENGTH CHIP

Contour length computation is useful for further processing such as structural saliency (Shaashua and Ullman, 1988), which is thought to be an important stage before object recognition. This computation is impossible on an analog chip if we

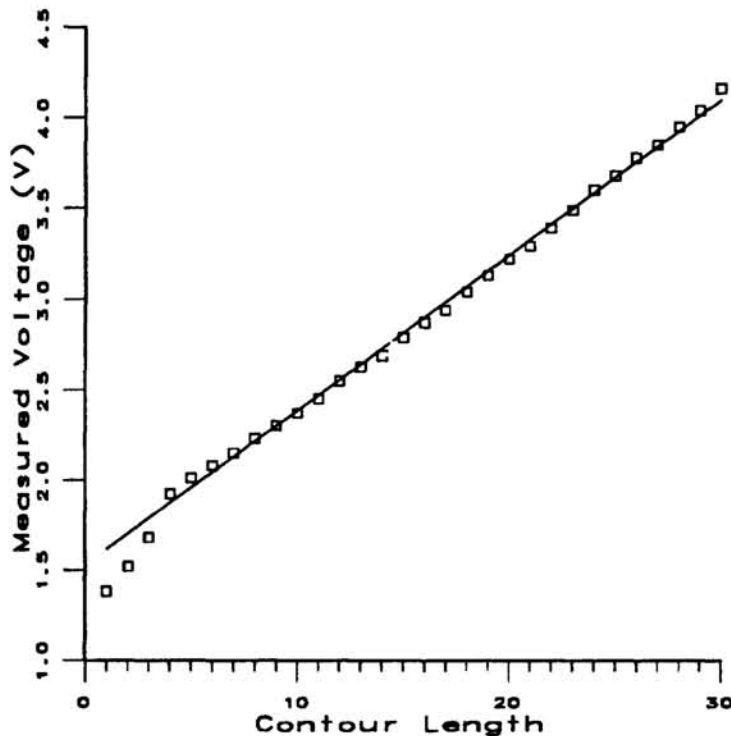

Figure 1: Plot of measured voltage vs. contour length from 30 different contours scanned into the contour length chip. The voltage is a linear function of contour length.

are restricted to pure pixel- or image-based operations. The *dynamic wire* methodology provides dedicated lines of communication among groups of pixels of an image which share common properties (Liu and Harris, 1992). In simple applications, object regions can be grouped together to compute the area or the center of mass of each object. Alternatively, object boundaries may be used to compute curvature or contour length. These ideas are not limited to sets of simple electrical wires; resistive networks can also be configured on the fly. The problem of smoothing object contours using resistive dynamic wires has been previously studied (Liu and Harris, 1992).

In the contour-length application, pixels along image contours are electrically connected by a reconfigurable dynamic wire. The first step of processing requires that each contour choose an arbitrary but unique *leader* pixel. The top of Fig. 2 shows several examples of contours and indicates which pixels where chosen as leaders by the chip. The leader is responsible for connecting a shunting resistor between the shared dynamic wire and ground. If each pixel on the contour supplies a constant amount of current to the dynamic wire, all of the current must flow through the shunting resistor. Therefore, the voltage on the wire will encode the contour length. Fig. 1 shows the linear relationship between the measured voltage and the contour length. The bottom half of Fig. 2 shows the length of several example contours using an intensity coding. The brighter contours indicate a higher voltage and therefore a longer contour. The contour length chip was fabricated through MOSIS using $2\mu m$ CMOS technology. The prototype 2x2 mm$^2$ chip contains an a 7x7 pixel array.

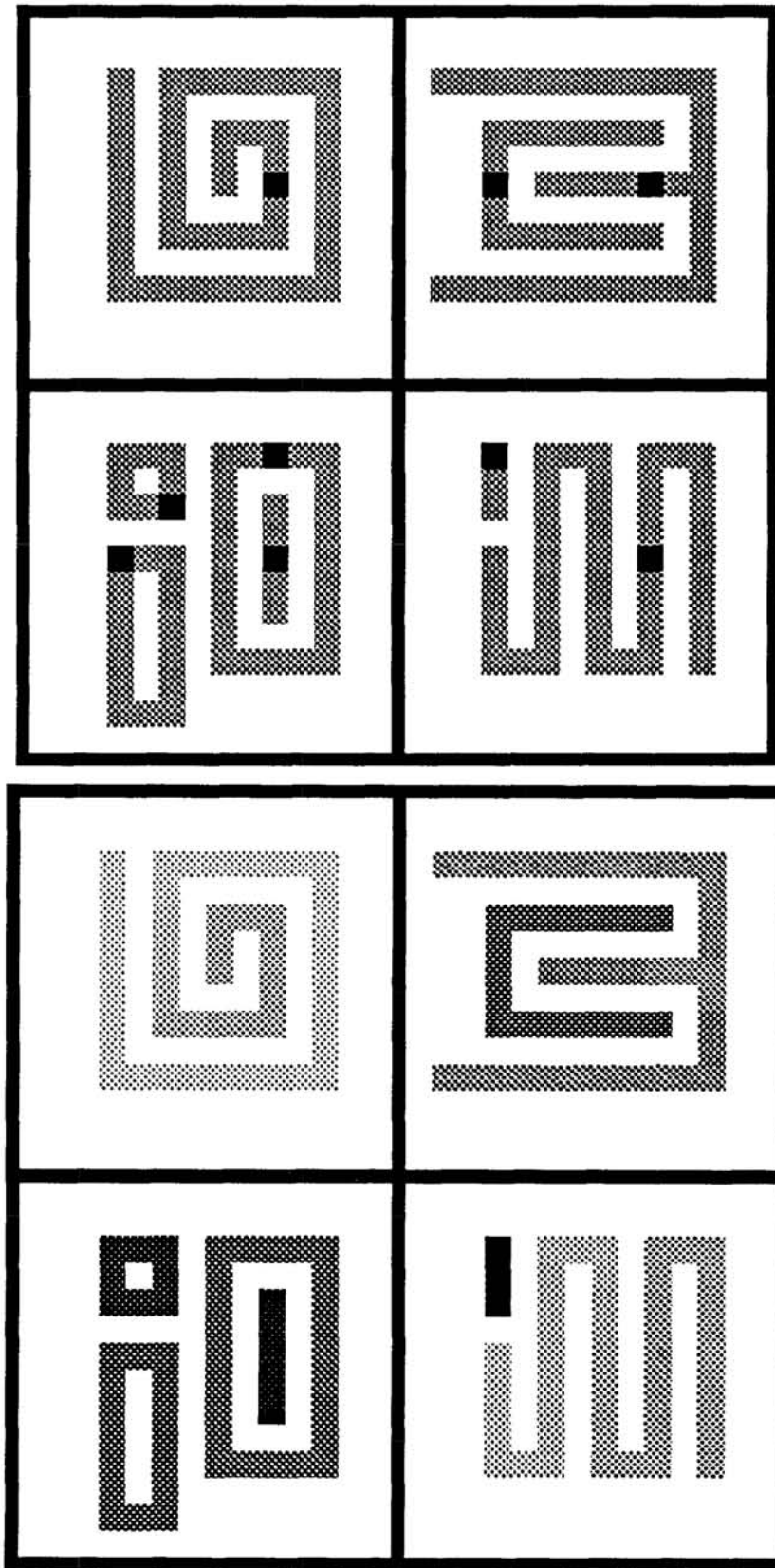

Figure 2: Four binary contour images were scanned into the contour-length chip and are shown in the top figure. The highlighted pixel in each contour was chosen by the chip to be the leader. The bottom figure shows the measured voltages (indicated by intensity) from the contour-length chip for the four images are shown. Since the intensity of each pixel encodes its length, the longer contours are brighter.

The most challenging aspect of the design of the contour-length chip is the circuitry to uniquely select a pixel from each contour to be the leader. The leader is selected by entering all the pixels along each contour in a competition. The winner of this competition will be the leader. This competition requires each node to charge up its own capacitor once a reset line has been triggered. The first node that charges its capacitor above the trip point of a digital inverter will pull down a global precharge wire which connects all the pixels along the contour. This wire will in turn latch the states of the winner and losers. One of the pixels will normally toggle first because of the inherent offsets and component mismatches in silicon.

## 2  FIGURE-GROUND CHIP

Ullman (1984) proposed that a visual routine is used in human vision to determine if a specified point in the visual field is inside or outside of one (or more) closed visual contours. We describe such a chip that labels all points inside a given— possibly incomplete and broken—contour. We assume that the presence of an edge in the image causes switches at the corresponding grid point within a rectangular resistive network to open (Fig. 3). A closed edge contour will then correspond to a series of open switches on this grid. We assume that the visual contour will always encompass the central grid point in the array. At this point, the resistive grid is connected to the battery $V_{fig}$, while the periphery of the array is grounded to $V_{gnd}$. If the voltage at all other grid points is left floating and the contour is complete, that is, the central grid point is completely isolated from the periphery of the chip by a series of open grid points, the voltage at all points inside the contour rises to $V_{fig}$, while the voltage at grid points outside the contour will settle to $V_{gnd}$. Thus, the *figure* will be labeled by one voltage level and *ground* by another.

Contours in real images are frequently incomplete, but instead have broken segments of one or more pixels. This will enable the current to flow through these holes in the contour, smearing out the voltage level between inside and outside. We exploit a property of Mead's (1989) Hres circuit, used to implement the resistances, to achieve contour completion. While the current flowing through Hres is linear in the voltage gradient for small voltage differences, it saturates for large voltage gradients. At those locations where the contour is broken, the saturating resistances limit the current flow, preventing smoothing of the voltage profile to occur.

Figure 4 shows the responses of the Figure-Ground chip to different input patterns collected with a fixed bias: $V_{fig} = 3.5\,V$ and $V_{gnd} = 2\,V$. The two-dimensional data is presented as pairs of images. The input patterns are located on the left while the corresponding voltage outputs are presented next to the input on the right. The black-white patterns are used to represent the binary input data encoding object boundaries. Thus, at all locations marked in black, the associated switches shown in Fig. 1a are opened. The gray-scale on the right denotes output voltage levels, where the darkest value corresponds to $V_{fig}$ and the brightest to $V_{gnd}$. The center pixel of the view field is always set to $V_{fig}$. Notice that at every node where a boundary input signal (in black) appears and the switches are opened, the output voltage at that node is tied to $V_{gnd}$. This can be seen best in (e; white outline). To evaluate the ability of our circuit to perform Figure-Ground segregation in the presence of breaks in the contour, more and wider breaks are introduced into a simple square

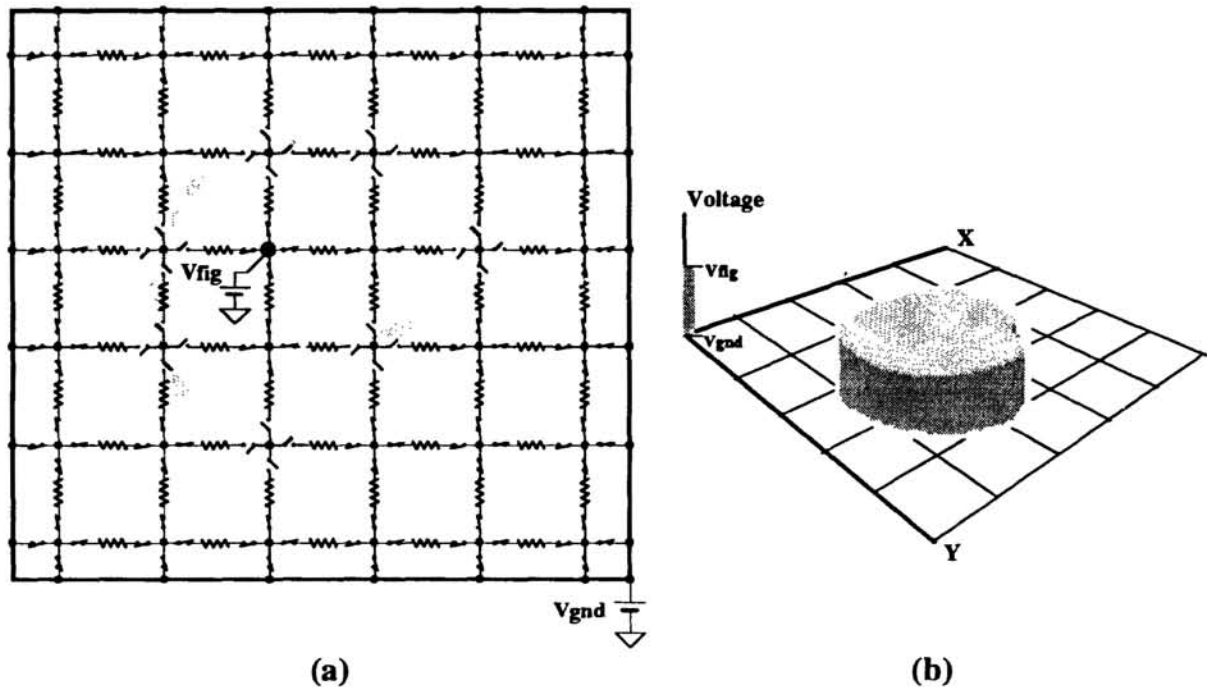

(a)                                          (b)

Figure 3: (a) The Figure-Ground network is made up of resistors and switches. The input to the chip is a binary edge map. At every grid point in the rectangular array where edges have been found, four switches are opened, isolating that node from its four neighbors (the shaded edge contour corresponds to a series of isolated nodes). We assume that the central point in the array is always enclosed by the contour. This point is connected to a voltage source $V_{fig}$, while the periphery is connected to the voltage $V_{gnd}$. If the contour is unbroken, the voltage at each interior point will then rise to $V_{fig}$, while all outside grid points will settle to $V_{gnd}$. Thus, the object is rapidly segregated from the background. If the contour is not complete, the saturating resistors (indicated with simple resistors) will limit the current flowing through these holes in the contour and partially seal off the boundary. (b) represents a conceptual view of how an object (figure) is segregated from the background in the two-dimensional view field, in terms of two distinct voltage levels ($V_{fig}$ labels the object and $V_{gnd}$ labels the background). The circuit has 48 by 48 nodes on a 4.6 by 6.8 $mm^2$ die size and was implemented using MOSIS 2 $\mu m$ CMOS technology.

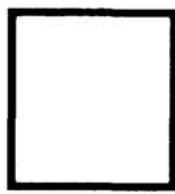 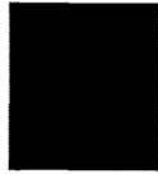

(a) The input consists of a completely enclosed box. The network is therefore broken into two isolated segments, the inside and the outside of the box and labeled by two very different voltage values, $V_{fig}$ and $V_{gnd}$.

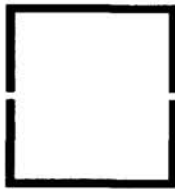 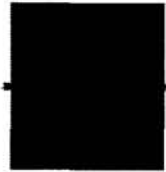

(b) The object boundary has a break equal to one pixel at the center of the left and right edges. Due to the large voltage difference across these two leaks, the saturated horizontal resistances, HRes, saturate, thereby helping to "seal" off these breaks using a very simple algorithm.

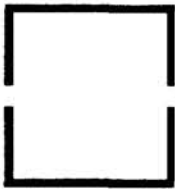 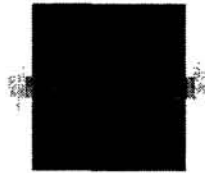

(c) The width of the breaks in the contour increases to three pixels each. Yet HRes still acts to effectively seal the two holes and the "Figure" is segregated from the "Surround".

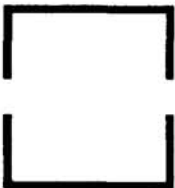 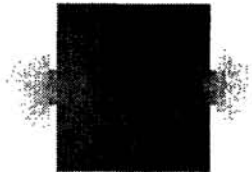

(d) The width of the breaks increases to five pixels each. Due to the much smaller voltage gradient across this wider gap in the contour, the voltage spreads outside the figure.

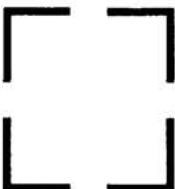 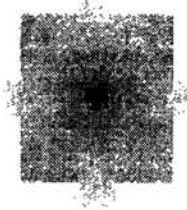

(e) A total of four breaks, each five pixels wide, prevents the "Figure" from being segregated. The system can't decide whether a **single object** with wide breaks at its side or **four separate objects** are present.

**Figure 4**

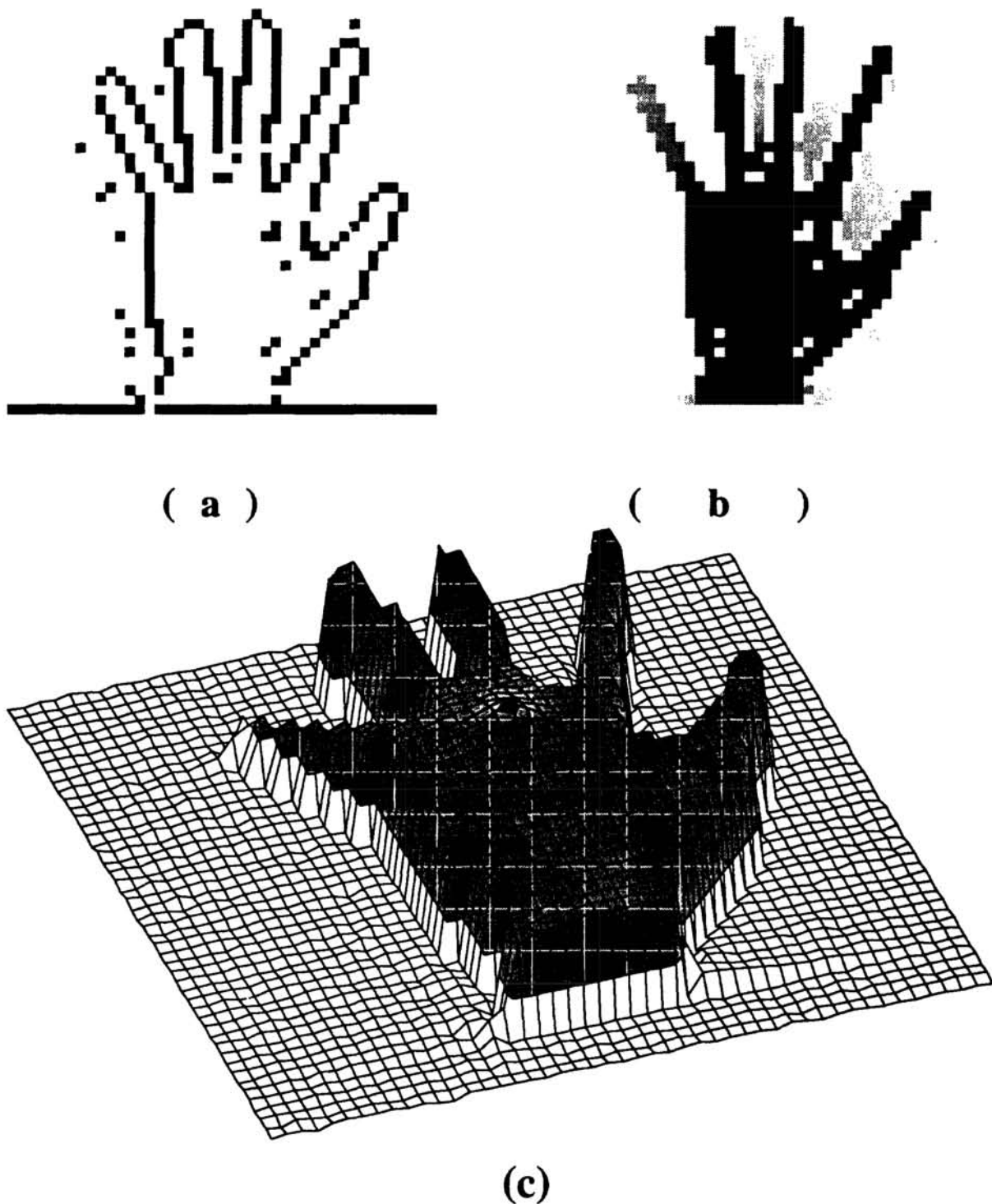

( a )                    ( b )

(c)

Figure 5: The Figure-Ground response to a noisy and incomplete contour outlining a hand (the binary image shown in (a) is scanned in from off-chip). The output voltage is shown as intensity in (b) and as a 3-D plot in (c). The center node is tied to 3.5 $V$ and marked as black in (c). The shaded area labels all pixels whose voltage is above 2.4 $V$. Notice the voltage decay along the little finger, due to an incomplete contour at the finger tip.

contour. The box and break points on the sides are center-row symmetrical and the breaks are respectively one, three and five pixels wide. In (e), two additional, five pixel wide breaks have been included. For small enough breaks, our circuit has an excellent boundary-completion capabilities. This is important for machine vision, since real images rarely have complete boundaries.

The performance of the chip is illustrated in Fig. 4. If the contour is unbroken, the voltage inside the figure rises to $V_{fig}$, segregating it from the surround. If a small gap appears in the contour, it can be partially sealed off by the action of the saturating resistance Hres, which limits the current flowing through this gap, inhibiting full voltage equalization from occurring. As the break in the contour becomes larger, the voltage gradient across the *illusionary contour* between the upper and the lower part of the figure becomes smaller and smaller. If Hres is set to a low conductance, the gradient becomes larger again (Fig. 5c); now, however, the chip fails to discriminate between very small and large gaps. Note that inside and outside are strictly defined only for a closed contour. Thus, it is somewhat arbitrary at what distance two edges are considered to be part of the same or separate contours (e.g., Fig. 5). If the output voltage is thresholded at 3.0 V (in the case of Fig. 5b), the contour with one or two pixel breaks would be considered a single Figure, while the two larger breaks would not be.

## 3  CONCLUSION

Most analog vision chips are restricted to work either at the local, pixel-level or the global, image-level. The dynamic wire and figure-ground chips discussed in this paper allow data-dependent neighborhoods to form. With these configured neighborhoods, analog chips can now perform object-level processing.

**Acknowledgements**

This Work is supported by the National Science Foundation, the Office of Naval Research and Rockwell International Science Center. We thank MOSIS for all chip fabrication. JGH is supported by an NSF postdoctoral fellowship.

**References**

Liu, S. and Harris, J.G. (1992), Dynamic wires: an analog VLSI model for object processing, *Internat. Journal of Comp. Vision.* 8: pp. 231-239.

Luo, J., Koch, C. and Mathur, B. (1992), Figure-Ground segregation using an analog VLSI Chip, *IEEE Micro*, Vol. 12 46-57, 1992.

Shaashua, A. and Ullman, S. (1988), Structural saliency: The detection of globally salient structures using a locally connected network. In *Proceedings of the IEEE Computer Vision and Pattern Recognition Conference.*

Ullman, S. (1984), Visual routines, *Cognition*, Vol. 18, pp. 97-159, 1984.